# The Power of Approximating: a Comparison of Activation Functions

**Bhaskar DasGupta**
Department of Computer Science
University of Minnesota
Minneapolis, MN 55455-0159
email: dasgupta@cs.umn.edu

**Georg Schnitger**
Department of Computer Science
The Pennsylvania State University
University Park, PA 16802
email: georg@cs.psu.edu

## Abstract

We compare activation functions in terms of the approximation power of their feedforward nets. We consider the case of analog as well as boolean input.

## 1 Introduction

We consider efficient approximations of a given multivariate function $f : [-1, 1]^m \to \mathcal{R}$ by feedforward neural networks. We first introduce the notion of a feedforward net.

Let $\Gamma$ be a class of real-valued functions, where each function is defined on some subset of $\mathcal{R}$. A $\Gamma$-net $C$ is an unbounded fan-in circuit whose edges and vertices are labeled by real numbers. The real number assigned to an edge (resp. vertex) is called its *weight* (resp. its *threshold*). Moreover, to each vertex $v$ an activation function $\gamma_v \in \Gamma$ is assigned. Finally, we assume that $C$ has a single sink $w$.

The net $C$ computes a function $f_C : [-1, 1]^m \to \mathcal{R}$ as follows. The components of the input vector $x = (x_1, \ldots, x_m) \in [-1, 1]^m$ are assigned to the sources of $C$. Let $v_1, \ldots, v_n$ be the immediate predecessors of a vertex $v$. The input for $v$ is then $s_v(x) = \sum_{i=1}^{n} w_i y_i - t_v$, where $w_i$ is the weight of the edge $(v_i, v)$, $t_v$ is the threshold of $v$ and $y_i$ is the value assigned to $v_i$. If $v$ is not the sink, then we assign the value $\gamma_v(s_v(x))$ to $v$. Otherwise we assign $s_v(x)$ to $v$.

Then $f_C = s_w$ is the function computed by $C$ where $w$ is the unique sink of $C$.

A great deal of work has been done showing that nets of two layers can approximate (in various norms) large function classes (including continuous functions) arbitrarily well (Arai, 1989; Carrol and Dickinson, 1989; Cybenko, 1989; Funahashi, 1989; Gallant and White, 1988; Hornik *et al.* 1989; Irie and Miyake,1988; Lapades and Farber, 1987; Nielson, 1989; Poggio and Girosi, 1989; Wei *et al.*, 1991). Various activation functions have been used, among others, *the cosine squasher, the standard sigmoid, radial basis functions, generalized radial basis functions, polynomials, trigonometric polynomials and binary thresholds*. Still, as we will see, these functions differ greatly in terms of their approximation power when we only consider efficient nets; i.e. nets with *few layers* and *few vertices*.

Our goal is to compare activation functions in terms of *efficiency* and *quality of approximation*. We measure efficiency by the *size* of the net (i.e. the number of vertices, not counting input units) and by its *number of layers*. Another resource of interest is *the Lipschitz-bound* of the net, which is a measure of the numerical stability of the net. We say that net $C$ has Lipschitz-bound $L$ if all weights and thresholds of $C$ are bounded in absolute value by $L$ and for each vertex $v$ of $C$ and for all inputs $x, y \in [-1, 1]^m$,

$$|\gamma_v(s_v(x)) - \gamma_v(s_v(y))| \leq L \cdot |s_v(x) - s_v(y)|.$$

(Thus we do not demand that activation function $\gamma_v$ has Lipschitz-bound $L$, but only that $\gamma_v$ has Lipschitz-bound $L$ for the inputs it receives.) We measure the quality of an approximation of function $f$ by function $f_C$ by the Chebychev norm; i.e. by the maximum distance between $f$ and $f_C$ over the input domain $[-1, 1]^m$.

Let $\Gamma$ be a class of activation functions. We are particularly interested in the following two questions.

• Given a function $f : [-1, 1]^m \to \mathcal{R}$, how well can we approximate $f$ by a $\Gamma$-net with $d$ layers, size $s$, and Lipschitz-bound $L$? Thus, we are particularly interested in the behavior of the approximation error $e(s, d)$ as a *function of size and number of layers*. This set-up allows us to investigate how much the approximation error decreases with increased size and/or number of layers.

• Given two classes of activation functions $\Gamma_1$ and $\Gamma_2$, when do $\Gamma_1$-nets and $\Gamma_2$-nets have *essentially* the same "approximation power" with respect to some error function $e(s, d)$?

We first formalize the notion of "essentially the same approximation power".

**Definition 1.1** *Let $e : \mathcal{N}^2 \to \mathcal{R}_+$ be a function. $\Gamma_1$ and $\Gamma_2$ are classes of activation functions.*

(a). *We say that $\Gamma_1$ simulates $\Gamma_2$ with respect to $e$ if and only if there is a constant $k$ such that for all functions $f : [-1, 1]^m \to \mathcal{R}$ with Lipschitz-bound $1/e(s, d)$,*

*if $f$ can be approximated by a $\Gamma_2$-net with $d$ layers, size $s$, Lipschitz-bound $2^s$ and approximation error $e(s, d)$, then $f$ can also be approximated with error $e(s, d)$ by a $\Gamma_1$-net with $k(d+1)$ layers, size $(s+1)^k$ and Lipschitz-bound $2^{s^k}$.*

(b). *We say that $\Gamma_1$ and $\Gamma_2$ are equivalent with respect to $e$ if and only if $\Gamma_2$ simulates $\Gamma_1$ with respect to $e$ and $\Gamma_1$ simulates $\Gamma_2$ with respect to $e$.*

In other words, when comparing the approximation power of activation functions, we allow size to increase polynomially and the number of layers to increase by a constant factor, but we insist on at least the same approximation error. Observe that we have linked the approximation error $e(s, d)$ and the Lipschitz-bound of the function to be approximated. The reason is that approximations of functions with high Lipschitz-bound "tend" to have an inversely proportional approximation error. Moreover observe that the Lipschitz-bounds of the involved nets are allowed to be exponential in the size of the net. We will see in section 3, that for some activation functions far smaller Lipschitz-bounds suffice.

Below we discuss our results. In section 2 we consider the case of tight approximations, i.e. $e(s, d) = 2^{-s}$. Then in section 3 the more relaxed error model $e(s, d) = s^{-d}$ is discussed. In section 4 we consider the computation of boolean functions and show that sigmoidal nets can be far more efficient than threshold-nets.

## 2    Equivalence of Activation Functions for Error $e(s, d) = 2^{-s}$

We obtain the following result.

**Theorem 2.1** *The following activation functions are equivalent with respect to error $e(s, d) = 2^{-s}$.*

- *the standard sigmoid $\sigma(x) = \frac{1}{1+\exp(-x)}$,*
- *any rational function which is not a polynomial,*
- *any root $x^\alpha$, provided $\alpha$ is not a natural number,*
- *the logarithm (for any base $b > 1$),*
- *$e^x$,*
- *the gaussian $e^{-x^2}$,*
- *the radial basis functions $(1 + x^2)^\alpha$, $\alpha < 1$, $\alpha \neq 0$*

Notable exceptions from the list of functions equivalent to the standard sigmoid are polynomials, trigonometric polynomials and splines. We do obtain an equivalence to the standard sigmoid by allowing splines of degree $s$ as activation functions for nets of size $s$. (We will always assume that splines are continuous with a single knot only.)

**Theorem 2.2** *Assume that $e(s, d) = 2^{-s}$. Then splines (of degree s for nets of size s) and the standard sigmoid are equivalent with respect to $e(s, d)$.*

**Remark 2.1**

(a) *Of course, the equivalence of spline-nets and $\{\sigma\}$-nets also holds for* **binary** *input. Since threshold-nets can add and multiply m m-bit numbers with constantly many layers and size polynomial in m (Reif, 1987), threshold-nets can efficiently approximate polynomials and splines.*

*Thus, we obtain that $\{\sigma\}$-nets with $d$ layers, size $s$ and Lipschitz-bound $L$ can be simulated by nets of binary thresholds. The number of layers of the simulating threshold-net will increase by a constant factor and its size will increase by a polynomial in $(s+n)\log(L)$, where $n$ is the number of input bits. (The inclusion of $n$ accounts for the additional increase in size when approximately computing a weighted sum by a threshold-net.)*

*(b) If we allow size to increase by a polynomial in $s + n$, then threshold-nets and $\{\sigma\}$-nets are actually equivalent with respect to error bound $2^{-s}$. This follows, since a threshold function can easily be implemented by a sigmoidal gate (Maass et al., 1991).*

*Thus, if we allow size to increase polynomially (in $s+n$) and the number of layers to increase by a constant factor, then $\{\sigma\}$-nets with weights that are at most exponential (in $s+n$) can be simulated by $\{\sigma\}$-nets with weights of size polynomial in $s$.*

$\{\sigma\}$-nets and threshold-nets (respectively nets of linear thresholds) are not equivalent for analog input. The same applies to polynomials, even if we allow polynomials of degree $s$ as activation function for nets of size $s$:

**Theorem 2.3**

**(a)** *Let $sq(x) = x^2$. If a net of linear splines (with $d$ layers and size $s$) approximates $sq(x)$ over the interval $[-1, 1]$, then its approximation error will be at least $s^{-O(d)}$.*

**(b)** *Let $abs(x) = |x|$. If a polynomial net with $d$ layers and size $s$ approximates $abs(x)$ over the interval $[-1, 1]$, then the approximation error will be at least $s^{-O(d)}$.*

We will see in Theorem 2.5 that the standard sigmoid (and hence any activation function listed in Theorem 2.1) is capable of approximating $sq(x)$ and $abs(x)$ with error at most $2^{-s}$ by constant-layer nets of size polynomial in $s$. Hence the standard sigmoid is properly stronger than linear splines and polynomials. Finally, we show that *sine* and the standard sigmoid are inequivalent with respect to error $2^{-s}$.

**Theorem 2.4** *The function $sine(Ax)$ can be approximated by a $\{\sigma\}$-net $C_A$ with $d$ layers, size $s = A^{O(1/d)}$ and error at most $s^{O(-d)}$. On the other hand, every $\{\sigma\}$-net with $d$ layers which approximates $sine(Ax)$ with error at most $\frac{1}{2}$, has to have size at least $A^{\Omega(1/d)}$.*

Below we sketch the proof of Theorem 2.1. The proof itself will actually be more instructive than the statement of Theorem 2.1. In particular, we will obtain a general criterion that allows us to decide whether a given activation function (or class of activation functions) has at least the approximation power of splines.

## 2.1   Activation Functions with the Approximation Power of Splines

Obviously, any activation function which can efficiently approximate polynomials and the binary threshold will be able to efficiently approximate splines. This follows since a spline can be approximated by the sum $p + t \cdot q$ with polynomials $p$ and $q$

and a binary threshold $t$. (Observe that we can approximate a product once we can approximately square: $(x + y)^2/2 - x^2/2 - y^2/2 = x \cdot y$.)

Firstly, we will see that any sufficiently smooth activation function is capable of approximating polynomials.

**Definition 2.1** *Let $\gamma : R \rightarrow R$ be a function. We call $\gamma$ suitable if and only if there exists real numbers $\alpha, \beta$ ($\alpha > 0$) and an integer $k$ such that*

(a) *$\gamma$ can be represented by the power series $\sum_{i=0}^{\infty} a_i(x - \beta)^i$ for all $x \in [-\alpha, \alpha]$. The coefficients are rationals of the form $a_i = \frac{P_i}{Q_i}$ with $|P_i|, |Q_i| \leq 2^{ki}$ (for $i > 1$).*

(b) *For each $i > 2$ there exists $j$ with $i \leq j \leq i^k$ and $a_j \neq 0$.*

**Proposition 2.1** *Assume that $\gamma$ is suitable with parameter $k$.*

*Then, over the domain $[-D, D]$, any degree $n$ polynomial $p$ can be approximated with error $\varepsilon$ by a $\{\gamma\}$-net $C_p$. $C_p$ has 2 layers and size $O(n^{2k})$; its weights are rational numbers whose numerator and denominator are bounded in absolute value by*

$$p_{max}(2 + D)^{poly(n)}||\gamma^{(N+1)}||_{[-\alpha,\alpha]}\frac{1}{\varepsilon}.$$

*Here we have assumed that the coefficients of $p$ are rational numbers with numerator and denominator bounded in absolute value by $p_{max}$.*

Thus, in order to have at least the approximation power of splines, a suitable activation function has to be able to approximate the binary threshold. This is achieved by the following function class,

**Definition 2.2** *Let $\Gamma$ be a class of activation functions and let $g : [1, \infty] \rightarrow R$ be a function.*

(a). *We say that $g$ is fast converging if and only if*

$$| g(x) - g(x + \varepsilon) | = O(\varepsilon/x^2) \text{ for } x \geq 1, \varepsilon \geq 0,$$

$$0 < \int_1^{\infty} g(u^2)du < \infty \text{ and } | \int_{2^N}^{\infty} g(u^2)du | = O(1/N) \text{ for all } N \geq 1.$$

(b). *We say that $\Gamma$ is powerful if and only if at least one function in $\Gamma$ is suitable and there is a fast converging function $g$ which can be approximated for all $s > 1$ (over the domain $[-2^s, 2^s]$) with error $2^{-s}$ by a $\{\Gamma\}$-net with a constant number of layers, size polynomial in $s$ and Lipschitz-bound $2^s$.*

Fast convergence can be checked easily for differentiable functions by applying the mean value theorem. Examples are $x^{-\alpha}$ for $\alpha \geq 1$, $exp(-x)$ and $\sigma(-x)$. Moreover, it is not difficult to show that each function mentioned in Theorem 2.1 is powerful. Hence Theorem 2.1 is a corollary of

**Theorem 2.5** *Assume that $\Gamma$ is powerful.*

(a) *$\Gamma$ simulates splines with respect to error $e(s, d) = 2^{-s}$.*

**(b)**  *Assume that each activation function in Γ can be approximated (over the domain $[-2^s, 2^s]$) with error $2^{-s}$ by a spline-net $N_s$ of size $s$ and with constantly many layers. Then Γ is equivalent to splines.*

**Remark 2.2**  *Obviously, $1/x$ is powerful. Therefore Theorem 2.5 implies that constant-layer $\{1/x\}$-nets of size $s$ approximate $abs(x) = |x|$ with error $2^{-s}$. The degree of the resulting rational function will be polynomial in $s$. Thus Theorem 2.5 generalizes Newman's approximation of the absolute value by rational functions. (Newman, 1964)*

# 3    Equivalence of Activation Functions for Error $s^{-d}$

The lower bounds in the previous section suggest that the relaxed error bound $e(s, d) = s^{-d}$ is of importance. Indeed, it will turn out that many non-trivial smooth activation functions lead to nets that simulate $\{\sigma\}$-nets, **provided** the number of input units is counted when determining the size of the net. (We will see in section 4, that linear splines and the standard sigmoid are not equivalent if the number of inputs is **not** counted). The concept of *threshold-property* will be crucial for us.

**Definition 3.1**  *Let Γ be a collection of activation functions. We say that Γ has the threshold-property if there is a constant $c$ such that the following two properties are satisfied for all $m > 1$.*

**(a)**  *For each $\gamma \in \Gamma$ there is a threshold-net $T_{\gamma,m}$ with $c$ layers and size $(s+m)^c$ which computes the binary representation of $\gamma'(x)$ where $|\gamma(x) - \gamma'(x)| \leq 2^{-m}$.*

*The input $x$ of $T_{\gamma,m}$ is given in binary and consists of $2m+1$ bits; $m$ bits describe the integral part of $x$, $m$ bits describe its fractional part and one bit indicates the sign. $s+m$ specifies the required number of output bits, i.e. $s = \lceil \log_2(\sup\{\gamma(x) : -2^{m+1} < x < 2^{m+1}\}) \rceil$.*

**(b)**  *There is a Γ-net with $c$ layers, size $m^c$ and Lipschitz bound $2^{m^c}$ which approximates the binary threshold over $D = [-1, 1] - [-1/m, 1/m]$ with error $1/m$.*

We can now state the main result of this section.

**Theorem 3.1**  *Assume that $e(s, d) = s^{-d}$.*

**(a)**  *Let Γ be a class of activation functions and assume that Γ has the threshold property. Then, $\sigma$ and Γ are equivalent with respect to $e$. Moreover, $\{\sigma\}$-nets only require weights and thresholds of absolute value at most $s$. (Observe that Γ-nets are allowed to have weights as large as $2^s$!)*

**(b)**  *If Γ and $\sigma$ are equivalent with respect to error $2^{-s}$, then Γ and $\sigma$ are equivalent with respect to error $s^{-d}$.*

**(c)**  *Additionally, the following classes are equivalent to $\{\sigma\}$-nets with respect to $e$. (We assume throughout that all coefficients, weights and thresholds are bounded by $2^s$ for nets of size $s$).*

• *polynomial nets (i.e. polynomials of degree $s$ appear as activation function for nets of size $s$),*

- $\{\gamma\}$-nets, where $\gamma$ is a suitable function and $\gamma$ satisfies part (a) of Definition 3.1. (This includes the sine-function.)

- nets of linear splines

The equivalence proof involves a first phase of extracting $O(d \log s)$ bits from the analog input. In a second phase, a binary computation is mimicked. The extraction process can be carried out with error $s^{-1}$ (over the domain $[-1, 1] - [-1/s, 1/s]$) once the binary threshold is approximated.

## 4 Computing boolean functions

As we have seen in Remark 2.1, the binary threshold (respectively linear splines) gains considerable power when *computing boolean functions* as compared to *approximating analog functions*. But sigmoidal nets will be far more powerful when only the number of neurons is counted and the number of input units is disregarded. For instance, sigmoidal nets are far more efficient for "squaring", i.e when computing:

$$M_n = \{(x, y) : x \in \{0, 1\}^n, y \in \{0, 1\}^{n^2} \text{ and } [x]^2 \geq [y]\} \quad (\text{where } [z] = \sum_i z_i).$$

**Theorem 4.1** *A threshold-net computing $M_n$ must have size at least $\Omega(\log n)$. But $M_n$ can be computed by a $\sigma$-net with constantly many gates.*

The previously best known separation of threshold-nets and sigmoidal-nets is due to Maass, Schnitger and Sontag (Maass *et al.*, 1991). But their result only applies to threshold-nets with at most two layers; our result holds without any restriction on the number of layers. Theorem 4.1 can be generalized to separate threshold-nets and 3-times differentiable activation functions, but this smoothness requirement is more severe than the one assumed in (Maass *et al.*, 1991).

## 5 Conclusions

Our results show that good approximation performance (for error $2^{-s}$) hinges on two properties, namely efficient approximation of polynomials and efficient approximation of the binary threshold. These two properties are shared by a quite large class of activation functions; i.e. powerful functions. Since (non-polynomial) rational functions are powerful, we were able to generalize Newman's approximation of $|x|$ by rational functions.

On the other hand, for a good approximation performance relative to the relaxed error bound $s^{-d}$ it is already sufficient to efficiently approximate the binary threshold. Consequently, the class of equivalent activation functions grows considerably (but only if the number of input units is counted). The standard sigmoid is distinguished in that its approximation performance scales with the error bound: if larger error is allowed, then smaller weights suffice.

Moreover, the standard sigmoid is actually more powerful than the binary threshold even when computing boolean functions. In particular, the standard sigmoid is able to take advantage of its (non-trivial) smoothness to allow for more efficient nets.

**Acknowledgements.**    We wish to thank R. Paturi, K. Y. Siu and V. P. Roy-chowdhury for helpful discussions. Special thanks go to W. Maass for suggesting this research, to E. Sontag for continued encouragement and very valuable advice and to J. Lambert for his never-ending patience.

The second author gratefully acknowledges partial support by NSF-CCR-9114545.

# References

Arai, W. (1989), Mapping abilities of three-layer networks, *in* "Proc. of the International Joint Conference on Neural Networks", pp. 419-423.

Carrol, S. M., and Dickinson, B. W. (1989), Construction of neural nets using the Radon Transform,*in* "Proc. of the International Joint Conference on Neural Networks", pp. 607-611.

Cybenko, G. (1989), Approximation by superposition of a sigmoidal function, *Mathematics of Control, Signals, and System*, 2, pp. 303-314.

Funahashi, K. (1989), On the approximate realization of continuous mappings by neural networks, *Neural Networks*, 2, pp. 183-192.

Gallant, A. R., and White, H. (1988), There exists a neural network that does not make avoidable mistakes, *in* "Proc. of the International Joint Conference on Neural Networks", pp. 657-664.

Hornik, K., Stinchcombe, M., and White, H. (1989), Multilayer Feedforward Networks are Universal Approximators, *Neural Networks*, 2, pp. 359-366.

Irie, B., and Miyake, S. (1988), Capabilities of the three-layered perceptrons, *in* "Proc. of the International Joint Conference on Neural Networks", pp. 641-648.

Lapades, A., and Farbar, R. (1987), How neural nets work, *in* "Advances in Neural Information Processing Systems", pp. 442-456.

Maass, W., Schnitger, G., and Sontag, E. (1991), On the computational power of sigmoid versus boolean threshold circuits, *in* "Proc. of the 32nd Annual Symp. on Foundations of Computer Science", pp. 767-776.

Newman, D. J. (1964), Rational approximation to $|x|$, *Michigan Math. Journal*, 11, pp. 11-14.

Hecht-Nielson, R. (1989), Theory of backpropagation neural networks, *in* "Proc. of the International Joint Conference on Neural Networks", pp. 593-611.

Poggio, T., and Girosi, F. (1989), A theory of networks for Approximation and learning, *Artificial Intelligence Memorandum*, no 1140.

Reif, J. H. (1987), On threshold circuits and polynomial computation, *in* "Proceedings of the 2nd Annual Structure in Complexity theory", pp. 118-123.

Wei, Z., Yinglin, Y., and Qing, J. (1991), Approximation property of multi-layer neural networks ( MLNN ) and its application in nonlinear simulation, *in* "Proc. of the International Joint Conference on Neural Networks", pp. 171-176.